# Convergence Properties of some Spike-Triggered Analysis Techniques

**Liam Paninski**
Center for Neural Science
New York University
New York, NY 10003
*liam@cns.nyu.edu*
http://www.cns.nyu.edu/~liam

## Abstract

We analyze the convergence properties of three spike-triggered data analysis techniques. All of our results are obtained in the setting of a (possibly multidimensional) linear-nonlinear (LN) cascade model for stimulus-driven neural activity. We start by giving exact rate of convergence results for the common spike-triggered average (STA) technique. Next, we analyze a spike-triggered covariance method, variants of which have been recently exploited successfully by Bialek, Simoncelli, and colleagues. These first two methods suffer from extraneous conditions on their convergence; therefore, we introduce an estimator for the LN model parameters which is designed to be consistent under general conditions. We provide an algorithm for the computation of this estimator and derive its rate of convergence. We close with a brief discussion of the efficiency of these estimators and an application to data recorded from the primary motor cortex of awake, behaving primates.

## 1 Introduction

Systems-level neuroscientists have a few favorite problems, the most prominent of which is the "what" part of the neural coding problem: what makes a given neuron in a particular part of the brain fire? In more technical language, we want to know about the conditional probability distributions $P(spike|X = x)$, the probability that our cell emits a spike, given that some observable signal $X$ in the world takes value $x$. Because data is expensive, neuroscientists typically postulate a functional form for this collection of conditional distributions, and then fit experimental data to these functional models, in lieu of attempting to directly estimate $P(spike|X = x)$ for each possible $x$. In this paper, we analyze one such phenomenological model whose popularity seems to be on the rise:

$$p(spike|\vec{x}) = f(<\vec{k}_1, \vec{x}>, <\vec{k}_2, \vec{x}>, \ldots, <\vec{k}_m, \vec{x}>). \qquad (1)$$

Here $f$ is some arbitrary nonconstant, $\Re^m$-measurable, $[0,1]$-valued function, and $\{k_i\}$ are some linearly independent elements of the dual space, $X'$, of some topological vector space, $X$ — the space of possible "input signals." Interpret $f$ as a regular

conditional distribution. Roughly, then, the neuron projects the signal $\vec{x}$ onto some $m$-dimensional subspace spanned by $\{\vec{k}_i\}_{1 \leq i \leq m}$ (call this subspace $K$), then looks up its probability of firing based only on this projection. This model is often called a "linear-nonlinear," or "LN," cascade model. It is also a probabilistic analog of a certain type of "Wiener cascade" model; this class of models has received extensive study in the systems identification literature. (Note that this model is not the same as a Volterra series model; these two classes of systems have very different uniform approximation properties.)

The LN model has two important features. First, the spike trains of the cell are given by a conditionally (inhomogeneous) Poisson process given $\vec{x}$; that is, there are no dynamics in this model beyond those induced by $\vec{x}$ and $K$. Second, equation (1) implies:

$$p(spike|\vec{x}) = p(spike|\vec{x} + \vec{y}) \ \forall \ y \perp K. \tag{2}$$

In other words, the conditional probability of firing is constant along (hyper)planes in the input space. (The natural generalization of this is a model for which these surfaces of constant firing probability are manifolds of low codimension; however, we will stick to the linear case here.) This model is semiparametric in the sense that it separates the problem of learning $p(spike|\vec{x})$ into two pieces: 1) learning the finite-dimensional parameter $K$, and 2) learning the infinite-dimensional parameter $f$. If $K$ is given, the problem of learning $f$ reduces to a density estimation problem, about which much is known. The problem of estimating $K$ seems to be less well-understood, and we focus primarily on this problem here.

We start with some notation. Let $N$, as usual, denote the number of available samples, drawn from the fixed stimulus distribution $p(\vec{x})$ (in practice, of course, the samples from $p(\vec{x})$ are not independent; for simplicity, we will stick to the i.i.d. case here, but most of our methods can be extended to the more general case). Then our basic results will take the following form:

$$E\left(\text{Error}(\hat{K})\right) \sim \alpha N^{-\lambda} + \beta, \tag{3}$$

as $N$ becomes large. The estimator $\hat{K}$ is a deterministic map taking $N$ observations of stimulus and spike data (where spikes are binary random variables, conditionally independent given the stimulus) into an estimate of the true underlying $K$:

$$\hat{K} : (X \times \{0,1\})^N \quad \rightarrow \quad \mathcal{G}_m(X) \tag{4}$$

$$(\vec{x}_N, s_N) \quad \rightarrow \quad \hat{K}(\vec{x}_N, s_N), \tag{5}$$

where $(\vec{x}_N, s_N)$ denotes the $N$-sample data. $\mathcal{G}_m(X)$ is the $m$-Grassmann manifold of $X$, the space of all $m$-dimensional subspaces of $X$; the natural error metric, then, is the geodesic distance on $\mathcal{G}_m(X)$ (the "canonical angle") between the true subspace $K$ and the estimated subspace $\hat{K}$. For brevity, we will present most of our results in the $m = 1$ case only; here the metric takes the simple form

$$\text{Error}(\hat{K}) \equiv \cos^{-1} \frac{< \hat{K}, \vec{k}_1 >}{||\hat{K}|| ||\vec{k}_1||}. \tag{6}$$

The scalar terms $\lambda$, $\alpha$, and $\beta$ in (3) each depend on $f$, $K$, and $p(\vec{x})$; $\lambda$ is a constant giving the order of magnitude of convergence (usually, but not always, equal to $1/2$), $\alpha$ gives the precise convergence rate, and $\beta$ gives the asymptotic error. We will be mostly concerned with giving exact values for $\alpha$ and $\lambda$, and simply indicating when $\beta$ is zero or positive (i.e., when $\hat{K}$ is consistent in probability or not, respectively). As usual, rate-of-convergence results clarify why a given estimator works well (in

the sense that a only a small number of samples is needed for reliable estimates) in certain cases and poorly (sometimes not at all) in others.

We will discuss three estimators here; the first two are well-known, while the third is novel, and is consistent under much more general conditions. The first part of the paper will indicate how to derive representation (3), including the constants $\alpha$, $\beta$, and $\lambda$, for these three estimators. In the final two sections, we discuss lower bounds on the convergence rates of any possible $K$-estimator (these kinds of bounds provide a rigorous measure of the difficulty of this estimation problem), and then give a brief illustration of the new estimator applied to data recorded in the primary motor cortex of awake, behaving monkeys.

## 2 Convergence rates

All three of the estimators considered here can be naturally written as "M-estimators," that is,

$$\hat{K}(\vec{x}_N, s_N) \equiv \operatorname{argmax}_{V \in \mathcal{G}_m(X)} M_{(\vec{x}_N, s_N)}(V),$$

for some data-dependent function $M_N \equiv M_{(\vec{x}_N, s_N)}$ on $\mathcal{G}_m(X)$. Most of the mathematical labor in this section comes down to an application of the standard "delta method" from the theory of M-estimators [5]: typically the data-dependent (i.e., random) functions $M_N$ converge in some suitable sense, as $N \to \infty$, to some limit function $M$. The asymptotics of the M-estimator are then reduced to a study of 1) the variability of $M_N$ around the limit $M$ and 2) the local differential structure of $M$ in a neighborhood of the true value of the underlying parameter $K$. This program can be carried out trivially for the first two estimators but is more interesting for the third (the first two require only the multivariate CLT; the third requires an infinite-dimensional CLT).

### 2.1 Spike-triggered averaging

The first estimator, the spike-triggered average, is classical and very intuitive: $\hat{K}_{STA}$ is defined as the sample mean of the spike-conditional stimulus distribution $p(\vec{x}|spike)$; since the spike signal is binary, this is the same as the cross-correlation between the spike and the stimulus signal. (We assume throughout, without loss of generality, that $p(\vec{x})$ is centered, that is, $E(\vec{x}) = 0$.) We will also consider the following "linear regression" modification:

$$\hat{K}_{LR} \equiv A\hat{K}_{STA},$$

where $A$ is an operator chosen to "divide out" correlations in the stimulus distribution $p(\vec{x})$ ($A$ is typically the (pseudo-) inverse of the stimulus correlation matrix, which we will denote as $\sigma^2(p(\vec{x}))$). The analysis for $\hat{K}_{STA}$ and $\hat{K}_{LR}$ depends only on a straightforward application of the multivariate central limit theorem (CLT).

We begin with necessary and sufficient conditions for consistency. We assume throughout this paper that the stimulus distribution $p(\vec{x})$ has finite second moments; this assumption seems entirely reasonable on physical grounds. Let $q$ be a random variable with distribution given by

$$P(q) \equiv p(<\vec{x}, \vec{k}_1 > |spike) = \frac{f(<\vec{x}, \vec{k}_1 >)p(<\vec{x}, \vec{k}_1 >)}{\int_{\Re} f(<\vec{x}, \vec{k}_1 >)p(<\vec{x}, \vec{k}_1 >)}, \tag{7}$$

with $f$ as defined in (1) and $p(<\vec{x}, \vec{k}_1 >)$ denoting the one-dimensional projection of $p(\vec{x})$. The expectation of this random variable exists by the finite-variance assump-

tion on $p(\vec{x})$. Finally, as usual, we say $p(\vec{x})$ is radially symmetric if $p(B) = p(UB)$ for all Borel sets $B$ and all unitary transformations $U$.

**Theorem 1 ($\beta(\hat{K}_{STA})$).** *If $p(\vec{x})$ (resp. $p(A^{1/2}\vec{x})$) is radially symmetric and $E(q) \neq 0$, then $\beta(\hat{K}_{STA}) = 0$ (resp. $\beta(\hat{K}_{LR}) = 0$). Conversely, if $p(\vec{x})$ is radially symmetric and $E(q) = 0$, then $\beta > 0$, and if $p(\vec{x})$ is not radially symmetric, then there exists an $f$ for which $\beta > 0$.*

(Note that $f$ is not required to be smooth, or even continuous.) The above sufficiency conditions seem to be somewhat well-known; for example, most of the sufficiency statement appeared (albeit in somewhat less precise form) in [1]. On the other hand, the converse is novel, to our knowledge, and is perhaps surprisingly stringent. The first part of the necessity statement will be obvious from the following discussion of $\alpha$ (and in fact appears implicitly in [1]), while the second part is a little harder, and seems to require (rather elementary) characteristic function techniques. The proof proceeds by showing that a distribution is symmetric iff it has the property that the conditional mean of $\vec{x}$ is zero on all planar "slices" $< \vec{k}, \vec{x} > \in B$ for some $\vec{k} \in X'$ and real Borel set $B$.

Next we have the rate of convergence:

**Theorem 2 ($\alpha(\hat{K}_{STA})$).** *Assume $p(\vec{x})$ is symmetric normal, with standard deviation $\sigma(p)$. If $\beta(\hat{K}_{STA}) = 0$, then $N^{1/2}(\hat{K}_{STA} - K)$ is asymptotically normal with mean zero (considered as a distribution on the tangent plane of $\mathcal{G}_m(X)$ at the true underlying value $K$), and*

$$\alpha = \frac{\sigma(p)}{|E(q)|}\sqrt{\dim X - 1}.$$

Thus the performance of the spike-triggered average scales directly with the dimension of the ambient space and inversely with $E(q)$, a measure of the asymmetry of the spike-triggered distribution along $k_1$. Note that we stated the result under the much stronger condition that $p(\vec{x})$ is Gaussian. In this case, the form of $\alpha$ becomes quite simple, depending on the nonlinearity $f$ only through $E(q)$. The general case is proven by identical methods but results in a slightly more complicated ($f$-dependent) term in place of $\sigma(p)$. The proof follows by applying the multivariate central limit theorem to the sample mean random vectors drawn i.i.d. from the spike-conditional stimulus distribution, $p(\vec{x}|spike)$. The proof also supplies the asymptotic distribution of $Error(\hat{K}_{STA})$ (a noncentral F), which might be useful for hypothesis testing. The details are quite easy once the mean of this distribution is identified (as in [1], under the above sufficiency conditions), and we skip them to save room for more interesting results.

One final note: in stating the above two results, we have been assuming implicitly that $K$ is one-dimensional (since $\hat{K}_{STA}$ clearly returns a single vector, that is, a one-dimensional subspace of $X$). Nevertheless, the two theorems extend easily to the more general case, after $Error(\hat{K}_{STA})$ is redefined to measure angles between $m-$ and $1-$dimensional subspaces. (Of course, now $E(\hat{K}_{STA})$ and $\lim_{N\to\infty} \hat{K}_{STA}$ depend strongly on the input distribution $p(\vec{x})$, even for radially symmetric $p(\vec{x})$; see, e.g., [3] for an analysis of a special case of this effect.)

## 2.2 Covariance-based methods

The next estimator was introduced in an effort to extend spike-triggered analysis to the $m > 1$ case (see, e.g., [3], and references therein). Where $\hat{K}_{STA}$ was based on

the first moment of the spike-conditional stimulus distribution $p(\vec{x}|spike)$, $\hat{K}_{CORR}$ is based on the second moment. We define

$$\hat{K}_{CORR} \equiv (\sigma^2)^{-1}\text{eig}(\hat{\Delta\sigma^2}),$$

where eig($A$) denotes the significantly non-zero eigenspace of the operator $A$, and $\hat{\Delta\sigma^2}$ is some estimate (typically the usual sample covariance estimate) of the "difference-covariance" matrix $\Delta\sigma^2$, defined by

$$\Delta\sigma^2 \equiv \sigma^2(p(\vec{x})) - \sigma^2(p(\vec{x}|spike)).$$

Again, we start with $\beta$:

**Theorem 3** ($\beta(\hat{K}_{CORR})$). *If $p(\vec{x})$ is Gaussian and*

$$Var_{p(\vec{x}|spike)}(<\vec{k},\vec{x}>) \neq Var_{p(\vec{x})}(<\vec{k},\vec{x}>) \; \forall \vec{k} \in E_K,$$

*for some orthogonal basis $E_K$ of $K$, then $\beta(\hat{K}_{CORR}) = 0$. Conversely, if $p(\vec{x})$ is Gaussian and the variance condition is not satisfied for $f$, then $\beta > 0$, and if $p(\vec{x})$ is non-Gaussian, then there exists an $f$ for which $\beta > 0$.*

As before, the sufficiency is fairly well-known, while the necessity appears to be novel and relies on characteristic function arguments. It is perhaps surprising that the conditions on $p$ for the consistency of this estimator are even stricter than for the spike-triggered average. The essential fact here turns out to be that a distribution is normal iff, after a suitable change of basis, the conditional variance on all planar "slices" of the distribution is constant.

We have, with Odelia Schwartz, developed a striking inconsistency example which is worth mentioning here:

**Example (Inconsistency of $\hat{K}_{CORR}$).** *There is a nonempty open set of nonconstant $f$ and radially symmetric $p(\vec{x})$ such that $\hat{K}_{CORR}$ is asymptotically orthogonal to $K$ almost surely as $N \to \infty$. (In fact, the $f$ and $p$ in this set can be taken to be infinitely differentiable.)*

The basic idea is that, for nonnormal $p$, the spike-triggered variance of $<\vec{v},\vec{x}>$ depends on $f$ even for $\vec{v} \perp \vec{k}$; we leave the details to the reader.

We can derive a similar rate of convergence for these covariance-based methods. To reduce the notational load, we state the result for $m = 1$ only; in this case, we can define $\lambda_{\Delta\sigma^2}$ to be the (unique and nonzero by assumption) eigenvalue of $\Delta\sigma^2$.

**Theorem 4** ($\alpha(\hat{K}_{CORR})$). *Assume $p(\vec{x})$ is independent normal. If $\beta(\hat{K}_{CORR}) = 0$, then $N^{1/2}(\hat{K}_{CORR} - K)$ is asymptotically normal with mean zero and*

$$\alpha = \frac{\sigma\sqrt{\sigma^2 - \lambda_{\Delta\sigma^2}}}{|\lambda_{\Delta\sigma^2}|}\sqrt{\dim X - 1}.$$

(Again, while $\lambda_{\Delta\sigma^2}$ will not be exactly zero in practice, it can often be small enough that the asymptotic error remains prohibitively large for physiologically reasonable values of $N$.) The proof proceeds by applying the multivariate central limit theorem to the covariance matrix estimator, then examining the first-order Taylor expansion of the eigenspace map at $\Delta\sigma^2$; see the longer draft of this paper at http://www.cns.nyu.edu/~liam for the more general statement and proof.

## 2.3 Empirical processes techniques

We have seen that the two most common $K$-estimators are not consistent in general; that is, the asymptotic error $\beta$ is bounded away from zero for many (non-pathological) combinations of $p(\vec{x})$, $f$, and $K$. We now introduce a new estimator for which $\beta = 0$ under very general conditions (without, say, any symmetry or normality assumptions on $p$ or any symmetry assumptions on $f$).

The basic idea is that $K\vec{x}$ is in a sense a sufficient statistic for $\vec{x}$ (that is, $\vec{x} - K\vec{x}$ — *spike* forms a Markov chain). The data processing inequality suggests that we could estimate $K$ by maximizing

$$M_N(V) \equiv D_\phi(q_N(<V, \vec{x}>, spike); q_N(<V, \vec{x}>)q_N(spike)),$$

where $D_\phi$ is a functional with suitable convexity properties, and $q_N$ is some estimate of $p$. For example, we could let $D_\phi$ be an information divergence and $q_N$ some kernel estimate, that is, a filtered version of the empirical measure

$$p_N \equiv \frac{1}{N} \sum_{i=1}^{N} \delta_i$$

(see [4] for an independent approach along these lines). This doesn't quite work, however, because the kernel induces an arbitrary scale; if this scale is larger than the natural scale of $f$ and $p(<V, \vec{x}>)$ for some $V$ but not others, our estimate will be biased away from $K$. Therefore, $D_\phi$ and $p_N$ have to be asymptotically scale-free in some sense.

The simplest approach is to let the kernel width tend to zero as $N$ becomes large; it is even possible to calculate the optimal rate of kernel shrinkage in $N$, depending on the smoothness of $f$. It also turns out to be helpful to use a bias-corrected version of $M_N(V)$; a standard jackknife correction is sufficient to obtain an estimator which converges at the standard $\sqrt{N}$ rate. We have:

**Theorem 5** $(\beta(\hat{K}_\phi))$**.** *If $p$ has a nonzero density with respect to Lebesgue measure, $f$ is not constant a.e., and the kernel width goes to zero more slowly than $N^{r-1}$, for some $r > 0$, then $\beta = 0$ for the kernel estimator $\hat{K}_\phi$.*

In other words, this new estimator $\hat{K}_\phi$ works for very general neurons $f$ and stimulus distributions $p$; in particular, $\hat{K}_\phi$ is suitable for application to natural signal data. Clearly, the condition on $f$ is minimal; we ask only that the neuron be tuned. The condition on $p$ is quite weak (and can be relaxed further); we are simply ensuring that we are sampling from all of $X$, and in particular, the part of $X$ on which the cell is tuned.

Next we have the rate of convergence; in the following, the "approximation error" measures the difference between the true information divergence $M_\phi(V)$ and its kernel-smoothed version, defined in the obvious way.

**Theorem 6** $(\gamma$ *and* $\alpha$ *for* $(\hat{K}_\phi))$**.** *If the approximation error is of order $a_N^r$, $r > 1$, then the jackknifed kernel or histogram versions of $\hat{K}_\phi$, with bandwidth $N^s$, $-1 < s < -1/r$, converge at an $N^{-1/2}$ rate. Moreover, $N^{1/2}(\hat{K}_\phi - K)$ is asymptotically normal, with mean zero and easily calculable $\alpha(\hat{K}_\phi)$.*

The methods follow, e.g., example 3.2.12 of [5] — basically, a generalization of the classical theorem on the asymptotic distribution of the maximum likelihood estimator in regular parametric families. Again, see the longer draft at

 for the precise definition of the approximation error and the full expression for $\alpha(\hat{K}_\phi)$.

We have developed an algorithm for the computation of $\text{argmax}_V M_N(V)$, and numerical results show that $\hat{K}_\phi$ can be competitive with spike-triggered average or covariance techniques even in cases in which $\beta(\hat{K}_{STA})$ and $\beta(\hat{K}_{CORR})$ are zero. We present a brief application of $\hat{K}_\phi$ in section 4.

## 3  Lower bounds

Lower bounds for convergence rates provide a rigorous measure of the difficulty of a given estimation problem, or of the efficiency of a given estimator. We give a few such results below. The first lower bound is local, in the sense that we assume that the true parameter is known *a priori* to be in some small neighborhood of parameter space. For simplicity, assume for the moment that $p(\vec{x})$ is radially symmetric. Recall that the Hellinger metric between any two densities is defined as (half of) the $L_2$ distance between the square roots of the densities.

**Theorem 7 (Local (Hellinger) lower bound).** *For simplicity, let $p$ be standard normal. For any fixed differentiable $f$, uniformly bounded away from $0$ and $1$ and with a uniformly bounded derivative $f'$, and any Hellinger ball $\mathcal{F}$ around the true parameter $(f, K)$,*

$$\liminf_{N \to \infty} N^{1/2} \inf_{\hat{K}} \sup_{\mathcal{F}} E(Error(\hat{K})) \geq \left( \sigma(p)(E_p(\frac{|f'|^2}{f(1-f)}))^{1/2} \right)^{-1} \sqrt{\dim X - 1}.$$

The second infimum above is taken over all possible estimators $\hat{K}$. The right-hand side plays the role of the inverse Fisher information in the Cramer-Rao bound and is derived using a similarly local analysis; see [2] for details.

Global bounds are more subtle. We want to prove something like:

$$\liminf_{N \to \infty} a_N \inf_{\hat{K}} \sup_{\mathcal{F}(\epsilon)} E(Error(\hat{K})) \geq C(\epsilon),$$

where $\mathcal{F}(\epsilon)$ is some large parameter set containing, say, all $K$ and all $f$ for which some relevant measure of tuning is greater than $\epsilon$, $a_N$ is the corresponding convergence rate, and $C(\epsilon)$ plays the role of $\alpha(\hat{K})$ from the previous sections. So far, our most interesting results in this direction are negative:

**Theorem 8 (Information divergences are poor indices of $K$-difficulty).** *Let $\mathcal{F}(\epsilon)$ be the set of all $(K, f)$ for which the $\phi$-divergence "information" between $\vec{x}$ and spike is greater than $\epsilon$, that is,*

$$D_\phi(p(K\vec{x}, spike); \ p(spike)p(K\vec{x})) > \epsilon.$$

*Then, for $\epsilon > 0$ small enough, for any putative convergence rate $a_N$,*

$$\liminf_{N \to \infty} a_N \inf_{\hat{K}} \sup_{\mathcal{F}(\epsilon)} E(Error(\hat{K})) = \infty.$$

In other words, strictly information-theoretic measures of tuning do not provide a useful index of the difficulty of the $K$-learning problem; the intuitive explanation of this result is that purely measure-theoretic distance functions, like $\phi$-divergences, ignore the topological and vector space structure of the underlying probability measures, and it is exactly this structure that determines the convergence rates of any efficient $K$-estimator. To put it more simply, the learnability of $K$ depends on the smoothness of $f$, just as we saw in the last section.

# 4 Application to primary motor cortex data

We have applied these new spike-triggered analysis techniques to data collected in the primary motor cortex (MI) of awake, behaving monkeys in an effort to elucidate the neural encoding of time-varying hand position signals in MI. This analysis has led to several interesting findings on the encoding properties of these neurons, with immediate applications to the design of neural prosthetic devices. Here, we have room to mention only one result: the relevant $K$ for MI cells appear to be largely one-dimensional. In other words, the conditional firing rate of these neurons, given a specific time-varying hand path, is well captured by the following model (Fig. 1): $p(spike|\vec{x}) = f(<\vec{k}_0, \vec{x}>)$, where $\vec{x}$ represents the two-dimensional hand position signal in a temporal neighborhood of the current time, $\vec{k}_0$ is a cell-specific affine functional, and $f$ is a cell-independent scalar function.

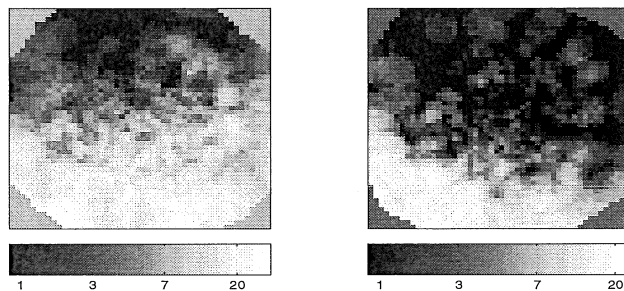

Figure 1: Example $\hat{f}(\hat{K}\vec{x})$ functions, computed from two different MI cells, with rank $\hat{K} = 2$; the x- and y-axes index $<\vec{k}_1, \vec{x}>$ and $<\vec{k}_2, \vec{x}>$, respectively, while the color axis indicates the value of $\hat{f}$ (the conditional firing rate given $K\vec{x}$), in Hz. The scale on the x- and y-axes is arbitrary and has been omitted. $\hat{K}$ was computed using the $\phi$-divergence estimator, and $\hat{f}$ was estimated using an adaptive kernel within the circular region shown (where sufficient data was available for reliable estimates). Note that the contours of this function are approximately linear; that is, $\hat{f}(\hat{K}\vec{x}) \approx f_0(<\vec{k}_0, \vec{x}>)$, where $\vec{k}_0$ is the vector orthogonal to the contour lines and $f_0$ is a suitably chosen scalar function on the line.

## Acknowledgements

We thank the Simoncelli lab for interesting discussions, and N. Rust and T. Sharpee for preliminary discussions of [4]. The MI experiments were done with M. Fellows, N. Hatsopoulos, and J. Donoghue. LP is supported by a HHMI predoctoral fellowship.

## References

[1] Chichilnisky, E. Network 12: 199-213 (2001).

[2] Gill, R. & Levit, B. Bernoulli, 1/2: 59-79 (1995).

[3] Schwartz, O., Chichilnisky, E. & Simoncelli, E. NIPS 14 (2002).

[4] Sharpee, T., Bialek, W. & Rust, N. This volume (2003).

[5] van der Vaart, A. & Wellner, J. Weak convergence and empirical processes. Springer-Verlag, New York (1996).
